# Relaxation Networks for Large Supervised Learning Problems

Joshua Alspector  Robert B. Allen  Anthony Jayakumar
Torsten Zeppenfeld and Ronny Meir
Bellcore
Morristown, NJ 07962-1910

## Abstract

Feedback connections are required so that the teacher signal on the output neurons can modify weights during supervised learning. Relaxation methods are needed for learning static patterns with full-time feedback connections. Feedback network learning techniques have not achieved wide popularity because of the still greater computational efficiency of back-propagation. We show by simulation that relaxation networks of the kind we are implementing in VLSI are capable of learning large problems just like back-propagation networks. A microchip incorporates deterministic mean-field theory learning as well as stochastic Boltzmann learning. A multiple-chip electronic system implementing these networks will make high-speed parallel learning in them feasible in the future.

## 1. INTRODUCTION

For supervised learning in neural networks, feedback connections are required so that the teacher signal on the output neurons can affect the learning in the network interior. Even though back-propagation[1] networks are feedforward in processing, they have implicit feedback paths during learning for error propagation. Networks with explicit, full-time feedback paths can perform pattern completion[2] and can have interesting temporal and dynamical properties in contrast to the single forward pass processing of multilayer perceptrons trained with back-propagation or other means. Because of the potential for complex dynamics, feedback networks require a reliable method of relaxation for learning and retrieval of static patterns. The Boltzmann machine[3] uses stochastic settling while the mean-field theory (MFT) version[4] [5] uses a more computationally efficient deterministic technique.

Neither of these feedback network learning techniques has achieved wide popularity because of the greater computational efficiency of back-propagation. However, this is likely to change in the near future because the feedback networks will be implemented in VLSI[6] making them available for learning experiments on high-speed parallel hardware.

In this paper, we therefore raise the following questions: whether these types of learning networks have the same representational and learning power as the more thoroughly studied back-propagation methods, how learning in such networks scales with problem size, and whether they can solve usefully large problems. Such questions are difficult to

answer with computer simulations because of the large amount of computer time required compared to back-propagation, but, as we show, the indications are promising.

## 2. SIMULATIONS

### 2.1 Procedure

In this section, we compare back-propagation, Boltzmann machine, and MFT networks on a variety of test problems. The back-propagation technique performs gradient descent in weight space by differentiation of an objective function, usually the error,

$$\varepsilon = \sum_{outputs\ k} (s_k^+ - s_k^-)^2$$

where $s_k^+$ is the target output and $s_k^-$ is the actual output. We choose to use the function

$$G = \sum_{outputs\ k} [s_k^+ \log(s_k^+/s_k^-) + (1-s_k^+)\log[(1-s_k^+)/(1-s_k^-)]] \tag{1}$$

for a more direct comparison to the Boltzmann machine[7] which has

$$G = \sum_{global\ states\ g} p_g^+ \log(p_g^+/p_g^-) \tag{2}$$

where $p_g$ is the probability of a global state.

Individual neurons in the Boltzmann machine have a probabilistic decision rule such that neuron $k$ is in state $s_k = 1$ with probability

$$p_i = \frac{1}{1+e^{-net_i/T}} \tag{3}$$

where $net_i = \sum_j w_{ij} s_j$ is the net input to each neuron and $T$ is a parameter that acts like temperature in a physical system and is represented by the noise term in Eq. (4), which follows. In the relaxation models, each neuron performs the activation computation

$$s_i = f(gain*(net_i + noise_i)) \tag{4}$$

where $f$ is a monotonic non-linear function such as *tanh*. In simulations of the Boltzmann machine, this is a step function corresponding to a high value of *gain*. The noise is chosen from a zero mean gaussian distribution whose width is proportional to the temperature. This closely approximates the distribution in Eq. (3) and matches our hardware implementation, which supplies uncorrelated noise to each neuron. The noise is slowly reduced as annealing proceeds. For MFT learning, the noise is zero but the gain term has a finite value proportional to $1/T$ taken from the annealing schedule. Thus the non-linearity sharpens as 'annealing' proceeds.

The network is annealed in two phases, + and −, corresponding to clamping the outputs in the desired state and allowing them to run free at each pattern presentation. The learning rule which adjusts the weights $w_{ij}$ from neuron $j$ to neuron $i$ is

$$\Delta w_{ij} = sgn[(s_i s_j)^+ - (s_i s_j)^-]. \tag{5}$$

Note that this measures the instantaneous correlations after annealing. For both phases each synapse memorizes the correlations measured at the end of the annealing cycle and weight adjustment is then made, (i.e., online). The *sgn* matches our hardware

implementation which changes weights by one each time.

## 2.2 Scaling

To study learning time as a function of problem size, we chose as benchmarks the parity and replication (identity) problems. The parity problem is the generalization of exclusive-OR for arbitrary input size, $n$. It is difficult because the classification regions are disjoint with every change of input bit, but it has only one output. The goal of the replication problem is for the output to duplicate the bit pattern found on the input after being transformed by the hidden layer. There are as many output neurons as input. For the replication problem, we chose the hidden layer to have the same number of neurons as the input layer, while for parity we chose the hidden layer to have twice the number as the input layer.

For back-propagation simulations, we used a learning rate of 0.3 and zero momentum. For MFT simulations, we started at a high temperature of $T_{hi} = K (1.4)^{10} \sqrt{(fanin)}$ where $K = 1\text{--}10$. We annealed in 20 steps dividing the temperature by 1.4 each time. The *fanin* parameter is the number of inputs from other neurons to a neuron in the hidden layer. We did 3 neuron update cycles at each temperature. For Boltzmann, we increased this to 11 updates because of the longer equilibration time. We used high gain rather than strictly binary units because of the possibility that the binary Boltzmann units would have exactly zero net input making annealing fruitless.

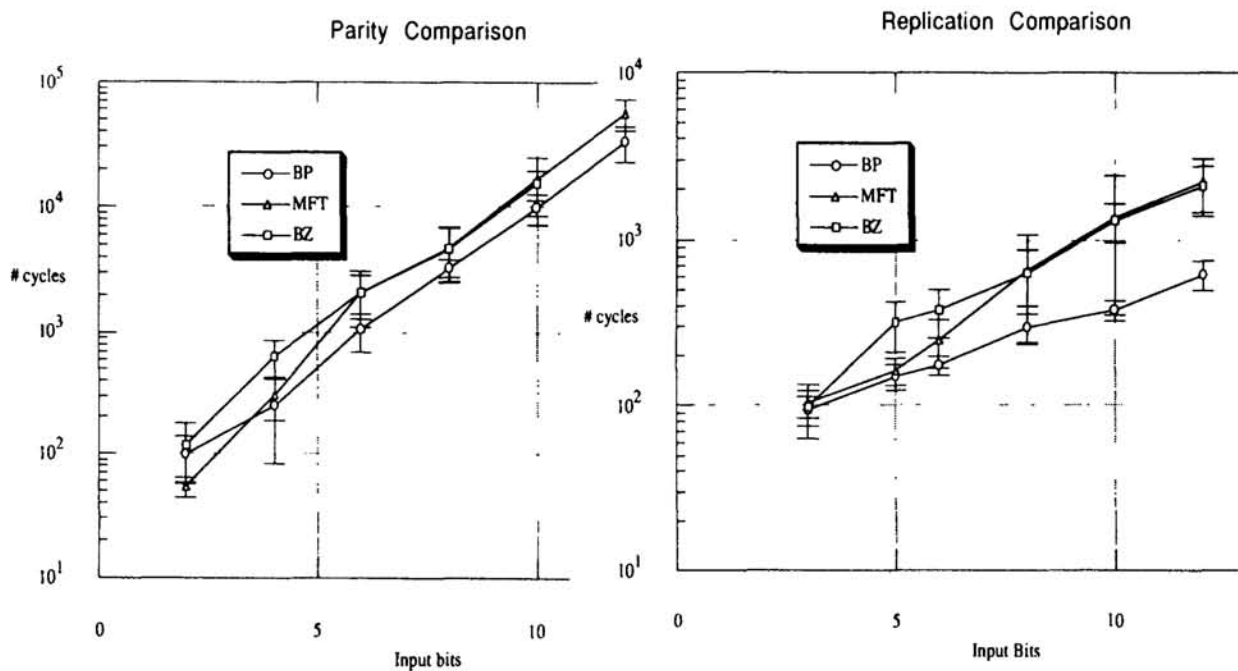

Figure 1. Scaling of Parity (1a) and Replication (1b) Problem with Input Size

Fig. 1a plots the results of an average of 10 runs and shows that the number of patterns required to learn to 90% correct for parity scales as an exponential in $n$ for all three networks. This is not surprising since the training set size is exponential and no constraints were imposed to help the network generalize from a small amount of data. An activation range of -1 to 1 was used on both this problem and the replication problem. There is no appreciable difference in learning as a function of patterns presented. Actual

computer time is larger by an additional factor of $n^2$ to account for the increase in the number of connections. Direct parallel implementation will reduce this additional factor to less than $n$. Computer time for MFT learning was an additional factor of 10 slower than back-propagation and stochastic Boltzmann learning was yet another factor of 10 slower. The hardware implementation will make these techniques roughly equal in speed and far faster than any simulation of back-propagation. Fig. 1b shows analogous results for the replication problem.

## 2.3 NETtalk

As an example of a large problem, we chose the NETtalk[8] corpus with 20,000 words. Fig. 2 shows the learning curves for back-propagation, Boltzmann, and MFT learning. An activation range of 0 to 1 gave the best results on this problem, possibly due to the sparse coding of text and phonemes. We can see that back-propagation does better on this problem which we believe may be due to the ambiguity in mapping letters to multiple phonemic outputs.

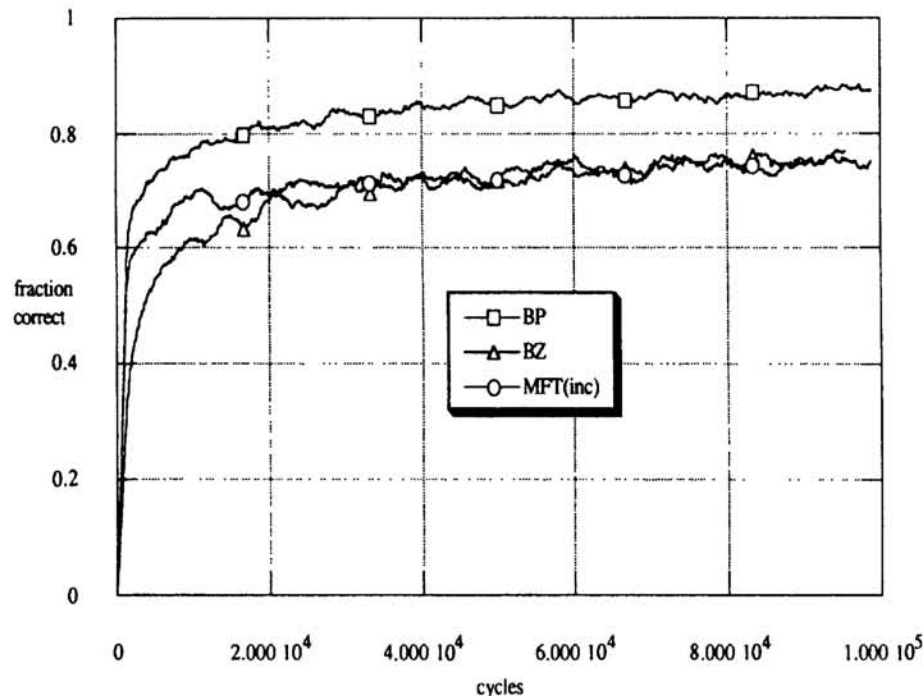

Figure 2. Learning Curves for NETtalk

## 2.4 Dynamic Range Manipulation

For all problems, we checked to see if reducing the dynamic range of the weights to 5 bits, equivalent to our VLSI implementation, would hinder learning. In most cases, there was no effect. Dynamic range was a limitation for the two largest replication problems with MFT. By adding an occasional global decay which decremented the absolute value of the weights, we were able to achieve good learning. Our implementation is capable of doing this. There was also a degradation of performance on the back-propagation version of the parity problem which took about a factor of three longer to learn with a 5 bit weight range.

## 3. VLSI IMPLEMENTATION

The previous section shows that relaxation networks are as capable as back-propagation networks of learning large problems even though they are slower in computer simulations. We are, however, implementing these feedback networks in VLSI which will speed up learning by many orders of magnitude. Our choice of learning technique for implementation is due mainly to the local learning rule which makes it much easier to cast these networks into electronics than back-propagation.

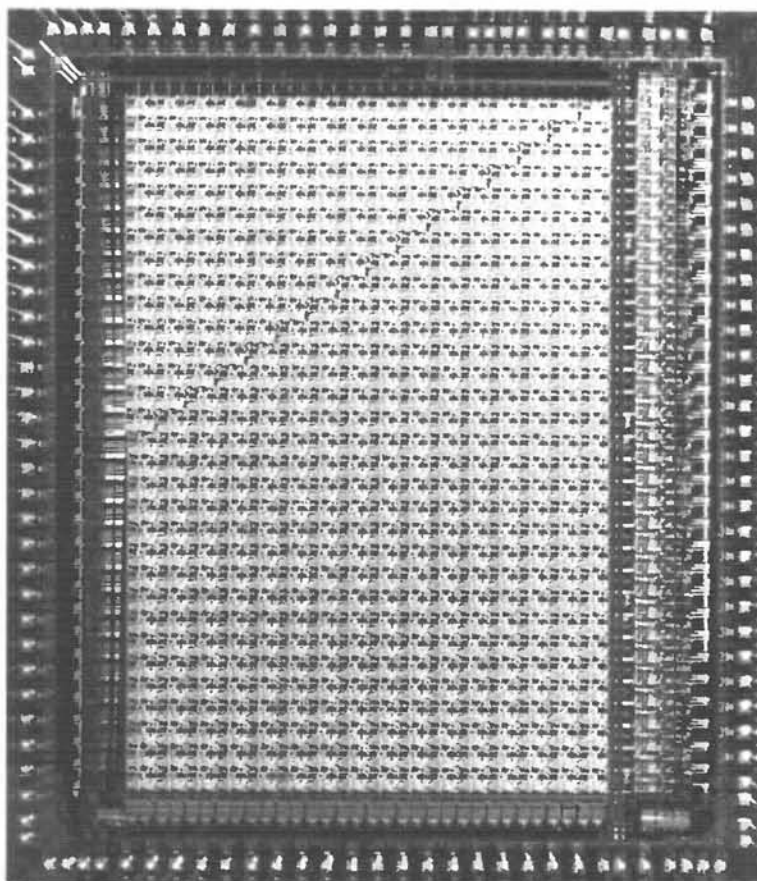

Figure 3. Photo of 32-Neuron Bellcore Learning Chip

Fig. 3 shows a microchip which has been fabricated. It contains 32 neurons and 992 connections (496 bidirectional synapses). On the extreme right is a noise generator which supplies 32 uncorrelated pseudo-random noise sources[9] to the neurons to their left. These noise sources are summed along with the weighted post-synaptic signals from other neurons at the input to each neuron in order to implement the simulated annealing process of the stochastic Boltzmann machine. The neuron amplifiers implement a non-linear activation function which has variable gain to provide for the gain sharpening function of the MFT technique. The range of neuron gain can also be adjusted to allow for scaling in summing currents due to adjustable network size.

Most of the area is occupied by the synapse array. Each synapse digitally stores a weight ranging from -15 to +15 as 4 bits plus a sign. It multiples the voltage input from the presynaptic neuron by this weight to output a current. One conductance direction can be disconnected so that we can experiment with asymmetric networks in accordance with our recent findings[10]. Although the synapses can have their weights set externally, they are designed to be adaptive. They store correlations using the local learning rule of Eq.

(5) and adjust their weights accordingly.

Although the chip is still being tested, some measurements can be reported. Fig. 4a shows a family of transfer functions of a neuron, showing how the gain is continually adjustable by varying a control voltage. Fig. 4b shows the transfer function of a synapse as different weights are loaded. The input linear range is about 2 volts.

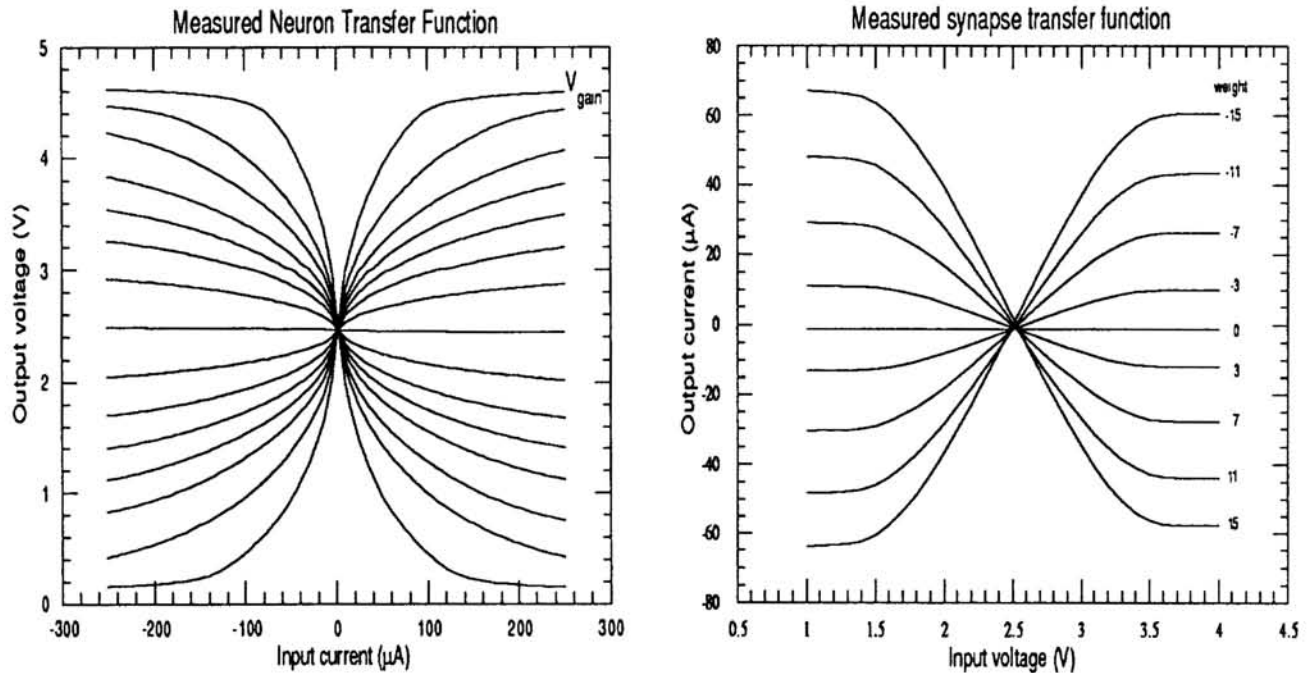

Figure 4. Transfer Functions of Electronic Neuron and Synapse

Fig. 5 shows two different neuron outputs with a decreasing noise signal added in. The upper trace shows a neuron driven by a function generator while the center trace shows an undriven neuron. The lower trace is the noise control voltage common to all neurons.

The chip is designed to be cascaded with other similar chips in a board-level system which can be accessed by a computer. The nodes which sum current from synapses for net input into a neuron are available externally for connection to other chips and for external clamping of neurons or other external input. We expect to be able to present roughly 100,000 patterns per second to the chip for learning as was determined from a previous prototype system[6] that was not cascadable. This speed will not be strongly affected by the increased network size of a multiple-chip system because of the inherent parallelism whereby each neuron and synapse updates its own state.

## 4. CONCLUSION

We have shown by simulation that relaxation networks of the kind we are implementing are as capable of learning large problems as back-propagation networks. A multiple-chip electronic system implementing these networks will make high-speed parallel learning in them feasible in the future.

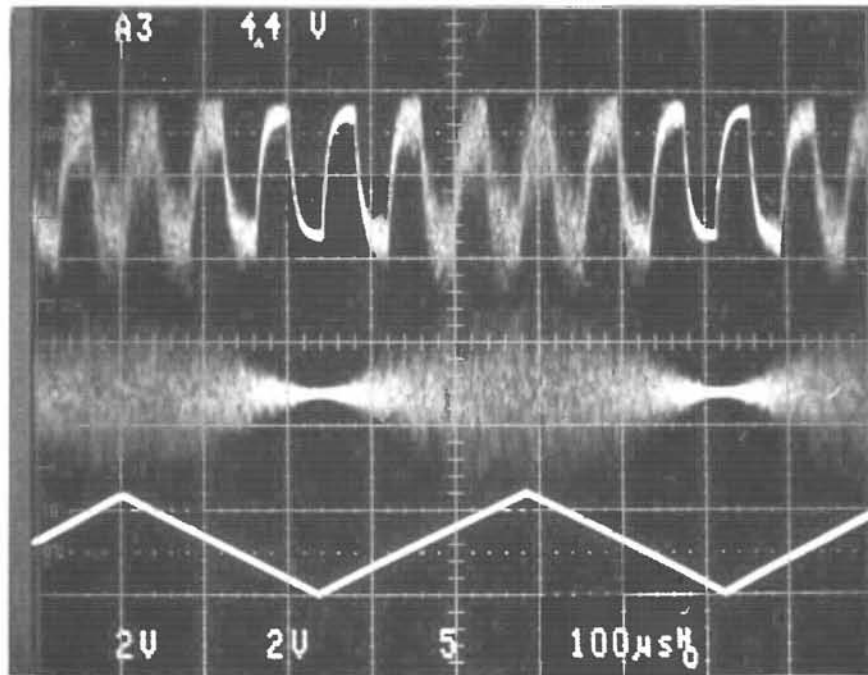

Figure 5.  Neuron Signals in the Presence of Noise Generator Input

*REFERENCES*

1. D.E. Rumelhart, G.E. Hinton, & R.J. Williams, "Learning Internal Representations by Error Propagation", in *Parallel Distributed Processing: Explorations in the Microstructure of Cognition. Vol. 1: Foundations*, D.E. Rumelhart & J.L. McClelland (eds.), MIT Press, Cambridge, MA (1986), p. 318.

2. J.J. Hopfield, "Neural Networks and Physical Systems with Emergent Collective Computational Abilities", *Proc. Natl. Acad. Sci. USA*, **79** , 2554-2558 (1982).

3. D.H. Ackley, G.E. Hinton, & T.J. Sejnowski, "A Learning Algorithm for Boltzmann Machines", *Cognitive Science* **9** (1985) pp. 147-169.

4. C. Peterson & J.R. Anderson, "A Mean Field Learning Algorithm for Neural Networks", *Complex Systems*, **1:5**, 995-1019, (1987).

5. G. Hinton, "Deterministic Boltzmann Learning Performs Steepest Descent in Weight-Space", Neural Computation, **1,** 143-150 (1989).

6. J. Alspector, B. Gupta, & R.B. Allen, "Performance of a Stochastic Learning Microchip" in *Advances in Neural Information Processing Systems* edited by D. Touretzky (Morgan-Kaufmann, Palo Alto), pp. 748-760. (1989).

7. J.J. Hopfield, "Learning Algorithms and Probability Distributions in Feed-Forward and Feed-Back networks", *Proc. Natl. Acad. Sci. USA*, **84**, 8429-8433 (1987).

8. T.J. Sejnowski & C.R. Rosenberg, "Parallel Networks that Learn to Pronounce English Text", *Complex Systems*, **1**, 145-168 (1987).

9. J. Alspector, J.W. Gannett, S. Haber, M.B. Parker, & R. Chu, "A VLSI-Efficient Technique for Generating Multiple Uncorrelated Noise Sources and Its Application to Stochastic Neural Networks", *IEEE Trans. Circuits & Systems*, **38**, 109, (Jan., 1991).

10. R.B. Allen & J. Alspector, "Learning of Stable States in Stochastic Asymmetric Networks", *IEEE Trans. Neural Networks,* **1,** 233-238, (1990).
